# Learning Local Error Bars
# for Nonlinear Regression

**David A. Nix**
Department of Computer Science
and Institute of Cognitive Science
University of Colorado
Boulder, CO 80309-0430
dnix@cs.colorado.edu

**Andreas S. Weigend**
Department of Computer Science
and Institute of Cognitive Science
University of Colorado
Boulder, CO 80309-0430
andreas@cs.colorado.edu *

## Abstract

We present a new method for obtaining local error bars for nonlinear regression, i.e., estimates of the confidence in predicted values that depend on the input. We approach this problem by applying a maximum-likelihood framework to an assumed distribution of errors. We demonstrate our method first on computer-generated data with locally varying, normally distributed target noise. We then apply it to laser data from the *Santa Fe Time Series Competition* where the underlying system noise is known quantization error and the error bars give local estimates of model misspecification. In both cases, the method also provides a weighted-regression effect that improves generalization performance.

## 1  Learning Local Error Bars Using a Maximum Likelihood Framework: Motivation, Concept, and Mechanics

Feed-forward artificial neural networks used for nonlinear regression can be interpreted as predicting the mean of the target distribution as a function of (conditioned on) the input pattern (e.g., Buntine & Weigend, 1991; Bishop, 1994), typically using one linear output unit per output variable. If parameterized, this conditional target distribution (CTD) may also be

This paper is available with figures in colors as ftp://ftp.cs.colorado.edu/pub/
Time-Series/MyPapers/nix.weigend_nips7.ps.Z .

viewed as an error model (Rumelhart *et al.*, 1995). Here, we present a simple method that provides higher-order information about the CTD than simply the mean. Such additional information could come from attempting to estimate the entire CTD with connectionist methods (e.g., "Mixture Density Networks," Bishop, 1994; "fractional binning,"Srivastava & Weigend, 1994) or with non-connectionist methods such as a Monte Carlo on a hidden Markov model (Fraser & Dimitriadis, 1994). While non-parametric estimates of the shape of a CTD require large quantities of data, our less data-hungry method (Weigend & Nix, 1994) assumes a specific parameterized form of the CTD (e.g., Gaussian) and gives us the value of the error bar (e.g., the width of the Gaussian) by finding those parameters which maximize the likelihood that the target data was generated by a particular network model. In this paper we derive the specific update rules for the Gaussian case. We would like to emphasize, however, that any parameterized unimodal distribution can be used for the CTD in the method presented here.[1]

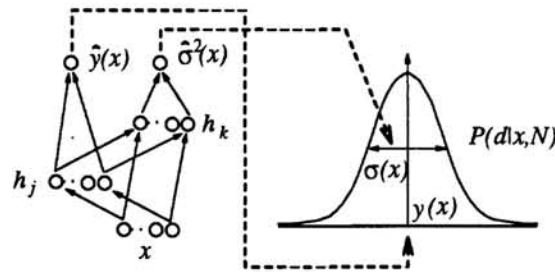

Figure 1: Architecture of the network for estimating error bars using an auxiliary output unit. All weight layers have full connectivity. This architecture allows the conditional variance $\hat{\sigma}^2$-unit access to both information in the input pattern itself and in the hidden unit representation formed while learning the conditional mean, $\hat{y}(\mathbf{x})$.

We model the desired observed target value $d$ as $d(\mathbf{x}) = y(\mathbf{x}) + n(\mathbf{x})$, where $y(\mathbf{x})$ is the underlying function we wish to approximate and $n(\mathbf{x})$ is noise drawn from the assumed CTD. Just as the conditional mean of this CTD, $y(\mathbf{x})$, is a function of the input, the variance $\sigma^2$ of the CTD, the noise level, may also vary as a function of the input $\mathbf{x}$ (noise heterogeneity). Therefore, not only do we want the network to learn a function $\hat{y}(\mathbf{x})$ that estimates the conditional mean $y(\mathbf{x})$ of the CTD, but we also want it to learn a function $\hat{\sigma}^2(\mathbf{x})$ that estimates the conditional variance $\sigma^2(\mathbf{x})$. We simply add an auxiliary output unit, the $\hat{\sigma}^2$-unit, to compute our estimate of $\sigma^2(\mathbf{x})$. Since $\sigma^2(\mathbf{x})$ must be positive, we choose an exponential activation function to naturally impose this bound: $\hat{\sigma}^2(\mathbf{x}) = \exp\left[\sum_k w_{\hat{\sigma}^2 k} h_k(\mathbf{x}) + \beta\right]$, where $\beta$ is the offset (or "bias"), and $w_{\hat{\sigma}^2 k}$ is the weight between hidden unit $k$ and the $\hat{\sigma}^2$-unit. The particular connectivity of our architecture (Figure 1), in which the $\hat{\sigma}^2$-unit has a hidden layer of its own that receives connections from both the $\hat{y}$-unit's hidden layer and the input pattern itself, allows great flexibility in learning $\hat{\sigma}^2(\mathbf{x})$. In contrast, if the $\hat{\sigma}^2$-unit has no hidden layer of its own, the $\hat{\sigma}^2$-unit is constrained to approximate $\sigma^2(\mathbf{x})$ using only the exponential of a linear combination of basis functions (hidden units) already tailored to represent $\hat{y}(\mathbf{x})$ (since learning the conditional variance $\hat{\sigma}^2(\mathbf{x})$ before learning the conditional mean $\hat{y}(\mathbf{x})$ is troublesome at best). Such limited connectivity can be too constraining on the functional forms for $\hat{\sigma}^2(\mathbf{x})$ and, in our experience,

produce inferior results. This is a significant difference compared to Bishop's (1994) Gaussian mixture approach in which all output units are directly connected to one set of hidden units. The other extreme would be not to share any hidden units at all, i.e., to employ two completely separate sets of hidden units, one to the $\hat{y}(\mathbf{x})$-unit, the other one to the $\hat{\sigma}^2(\mathbf{x})$-unit. This is the right thing to do if there is indeed no overlap in the mapping from the inputs to $y$ and from the inputs to $\sigma^2$. The two examples discussed in this paper are between these two extremes; this justifies the mixed architecture we use. Further discussion on shared vs. separate hidden units for the second example of the laser data is given by Kazlas & Weigend (1995, this volume).

For one of our network outputs, the $\hat{y}$-unit, the target is easily available—it is simply given by $d$. But what is the target for the $\hat{\sigma}^2$-unit? By maximizing the likelihood of our network model $\mathcal{N}$ given the data, $P(\mathcal{N}|\mathbf{x}, d)$, a target is "invented" as follows. Applying Bayes' rule and assuming statistical independence of the errors, we equivalently do gradient descent in the negative log likelihood of the targets $d$ given the inputs and the network model, summed over all patterns $i$ (see Rumelhart *et al.*, 1995): $C = -\sum_i \ln P(d_i|\mathbf{x}_i, \mathcal{N})$. Traditionally, the resulting form of this cost function involves only the estimate $\hat{y}(\mathbf{x}_i)$ of the conditional mean; the variance of the CTD is assumed to be constant for all $\mathbf{x}_i$, and the constant terms drop out after differentiation. In contrast, we allow the conditional variance to depend on $\mathbf{x}$ and explicitly keep these terms in $C$, approximating the conditional variance for $\mathbf{x}_i$ by $\hat{\sigma}^2(\mathbf{x}_i)$. Given any network architecture and any parametric form for the CTD (i.e., any error model), the appropriate weight-update equations for gradient decent learning can be straightforwardly derived.

Assuming normally distributed errors around $y(\mathbf{x})$ corresponds to a CTD density function of $P(d_i|\mathbf{x}_i) = [2\pi\sigma^2(\mathbf{x}_i)]^{-1/2} \exp\left\{-\frac{[d_i - y(\mathbf{x}_i)]^2}{2\sigma^2(\mathbf{x}_i)}\right\}$. Using the network output $\hat{y}(\mathbf{x}_i) \approx y(\mathbf{x}_i)$ to estimate the conditional mean and using the auxiliary output $\hat{\sigma}^2(\mathbf{x}_i) \approx \sigma^2(\mathbf{x}_i)$ to estimate the conditional variance, we obtain the monotonically related negative log likelihood, $-\ln P(d_i|\mathbf{x}_i, \mathcal{N}) = \frac{1}{2}\ln 2\pi\hat{\sigma}^2(\mathbf{x}_i) + \frac{[d_i - \hat{y}(\mathbf{x}_i)]^2}{2\hat{\sigma}^2(\mathbf{x}_i)}$. Summation over all patterns gives the total cost:

$$C = \frac{1}{2}\sum_i \left\{ \frac{[d_i - y(\mathbf{x}_i)]^2}{\hat{\sigma}^2(\mathbf{x}_i)} + \ln\hat{\sigma}^2(\mathbf{x}_i) + \ln 2\pi \right\} . \tag{1}$$

To write explicit weight-update equations, we must specify the network unit transfer functions. Here we choose a linear activation function for the $\hat{y}$-unit, tanh functions for the hidden units, and an exponential function for the $\hat{\sigma}^2$-unit. We can then take derivatives of the cost $C$ with respect to the network weights. To update weights connected to the $\hat{y}$ and $\hat{\sigma}^2$-units we have:

$$\Delta w_{\hat{y}j} = \eta \frac{1}{\hat{\sigma}^2(\mathbf{x}_i)}[d_i - \hat{y}(\mathbf{x}_i)]\, h_j(\mathbf{x}_i) \tag{2}$$

$$\Delta w_{\hat{\sigma}^2 k} = \eta \frac{1}{2\hat{\sigma}^2(\mathbf{x}_i)} \left\{ [d_i - \hat{y}(\mathbf{x}_i)]^2 - \hat{\sigma}^2(\mathbf{x}_i) \right\} h_k(\mathbf{x}_i) \tag{3}$$

where $\eta$ is the learning rate. For weights not connected to the output, the weight-update equations are derived using the chain rule in the same way as in standard backpropagation. Note that Eq. (3) is equivalent to training a separate function-approximation network for $\hat{\sigma}^2(\mathbf{x})$ where the targets are the squared errors $[d_i - y(\mathbf{x}_i)]^2]$. Note also that if $\hat{\sigma}^2(\mathbf{x}_i)$ is

constant, Eqs. (1)–(2) reduce to their familiar forms for standard backpropagation with a sum-squared error cost function.

The $1/\hat{\sigma}^2(\mathbf{x})$ term in Eqs. (2)–(3) can be interpreted as a form of "weighted regression," increasing the effective learning rate in low-noise regions and reducing it in high-noise regions. As a result, the network emphasizes obtaining small errors on those patterns where it can (low $\hat{\sigma}^2$); it discounts learning patterns for which the expected error is going to be large anyway (large $\hat{\sigma}^2$). This weighted-regression term can itself be highly beneficial where outliers (i.e., samples from high-noise regions) would ordinarily pull network resources away from fitting low-noise regions which would otherwise be well approximated.

For simplicity, we use simple gradient descent learning for training. Other nonlinear minimization techniques could be applied, however, but only if the following problem is avoided. If the weighted-regression term described above is allowed a significant influence *early* in learning, local minima frequently result. This is because input patterns for which low errors are initially obtained are interpreted as "low noise" in Eqs. (2)–(3) and overemphasized in learning. Conversely, patterns for which large errors are initially obtained (because significant learning of $\hat{y}$ has not yet taken place) are erroneously discounted as being in "high-noise" regions and little subsequent learning takes place for these patterns, leading to highly-suboptimal solutions. This problem can be avoided if we separate training into the following three phases:

**Phase I (Initial estimate of the conditional mean):** Randomly split the available data into equal halves, sets $\mathcal{A}$ and $\mathcal{B}$. Assuming $\sigma^2(\mathbf{x})$ is constant, learn the estimate of the conditional mean $\hat{y}(\mathbf{x})$ using set $\mathcal{A}$ as the training set. This corresponds to "traditional" training using gradient descent on a simple squared-error cost function, i.e., Eqs. (1)–(2) *without* the $1/\hat{\sigma}^2(\mathbf{x})$ terms. To reduce overfitting, training is considered complete at the minimum of the squared error on the cross-validation set $\mathcal{B}$, monitored at the end of each complete pass through the training data.

**Phase II (Initial estimate of the conditional variance):** Attach a layer of hidden units connected to both the inputs and the hidden units of the network from Phase I (see Figure 1). Freeze the weights trained in Phase I, and train the $\hat{\sigma}^2$-unit to predict the *squared errors* (see Eq. (3)), again using simple gradient descent as in Phase I. The training set for this phase is set $\mathcal{B}$, with set $\mathcal{A}$ used for cross-validation. If set $\mathcal{A}$ were used as the training set in this phase as well, any overfitting in Phase I could result in seriously underestimating $\sigma^2(\mathbf{x})$. To avoid this risk, we interchange the data sets. The initial value for the offset $\beta$ of the $\hat{\sigma}^2$-unit is the natural logarithm of the mean squared error (from Phase I) of set $\mathcal{B}$. Phase II stops when the squared error on set $\mathcal{A}$ levels off or starts to increase.

**Phase III (Weighted regression):** Re-split the available data into two new halves, $\mathcal{A}'$ and $\mathcal{B}'$. Unfreeze all weights and train all network parameters to minimize the full cost function $C$ on set $\mathcal{A}'$. Training is considered complete when $C$ has reached its minimum on set $\mathcal{B}'$.

## 2   Examples

**Example #1:** To demonstrate this method, we construct a one-dimensional example problem where $y(x)$ and $\sigma^2(x)$ are known. We take the equation $y(x) = \sin(\omega_\alpha x)\sin(\omega_\beta x)$ with $\omega_\alpha = 3$ and $\omega_\beta = 5$. We then generate $(x, d)$ pairs by picking $x$ uniformly from the interval $[0, \pi/2]$ and obtaining the corresponding target $d$ by adding normally distributed noise $n(x) = N[0, \sigma^2(x)]$ to the underlying $y(x)$, where $\sigma^2(x) = 0.02 + 0.25 \times [1 - \sin(\omega_\beta x)]^2$.

Table 1: Results for Example #1. $E_{NMS}$ denotes the mean squared error divided by the overall variance of the target; "Mean cost" represents the cost function (Eq. (1)) averaged over all patterns. Row 4 lists these values for the ideal model (true $y(x)$ and $\sigma^2(x)$) given the data generated. Row 5 gives the correlation coefficient between the network's predictions for the standard error (i.e., the square root of the $\hat{\sigma}^2$-unit's activation) and the actually occurring L1 residual errors, $|d(x_i) - \hat{y}(x_i)|$. Row 6 gives the correlation between the true $\sigma(x)$ and these residual errors. Rows 7–9 give the percentage of residuals smaller than one and two standard deviations for the obtained and ideal models as well as for an exact Gaussian.

| row | | Training | $(N = 10^3)$ | Evaluation | $(N = 10^5)$ |
|---|---|---|---|---|---|
| | | $E_{NMS}$ | Mean cost | $E_{NMS}$ | Mean cost |
| 1 | Phase I | 0.576 | 0.853 | 0.593 | 0.882 |
| 2 | Phase II | 0.576 | 0.542 | 0.593 | 0.566 |
| 3 | Phase III | 0.552 | 0.440 | 0.570 | 0.462 |
| 4 | $n(x)$ *(exact additive noise)* | 0.545 | 0.430 | 0.563 | 0.441 |
| | | $\rho$ | | $\rho$ | |
| 5 | $\rho(\hat{\sigma}(x)$, residual errors) | 0.564 | | 0.548 | |
| 6 | $\rho(\sigma(x)$, residual errors) | 0.602 | | 0.584 | |
| | | 1 std | 2 std | 1 std | 2 std |
| 7 | % of errors $< \hat{\sigma}(x); 2\hat{\sigma}(x)$ | 64.8 | 95.4 | 67.0 | 94.6 |
| 8 | % of errors $< \sigma(x); 2\sigma(x)$ | 66.6 | 96.0 | 68.4 | 95.4 |
| 9 | *(exact Gaussian)* | 68.3 | 95.4 | 68.3 | 95.4 |

We generate 1000 patterns for training and an additional $10^5$ patterns for post-training evaluation.

Training follows exactly the three phases described above with the following details:[2] Phase I uses a network with one hidden layer of 10 tanh units and $\eta = 10^{-2}$. For Phase II we add an auxiliary layer of 10 tanh hidden units connected to the $\hat{\sigma}^2$-unit (see Figure 1) and use the same $\eta$. Finally, in Phase III the composite network is trained with $\eta = 10^{-4}$.

At the end of Phase I (Figure 2a), the only available estimate of $\sigma^2(x)$ is the *global* root-mean-squared error on the available data, and the model misspecification is roughly uniform over $x$—a typical solution were we training with only the traditional squared-error cost function. The corresponding error measures are listed in Table 1. At the end of Phase II, however, we have obtained an initial estimate of $\sigma^2(x)$ (since the weights to the $\hat{y}$-unit are frozen during this phase, no modification of $\hat{y}$ is made). Finally, at the end of Phase III, we have better estimates of both $y(x)$ and $\sigma^2(x)$. First we note that the correlations between the predicted errors and actual errors listed in Table 1 underscore the near-optimal prediction of local errors. We also see that these errors correspond, as expected, to the assumed Gaussian error model. Second, we note that not only has the value of the cost function dropped from Phase II to Phase III, but *the generalization error has also dropped*, indicating an improved estimate of $y(x)$. By comparing Phases I and III we see that the quality of $\hat{y}(x)$ has improved significantly in the low-noise regions (roughly $x < 0.6$) at a minor sacrifice of accuracy in the high-noise region.

**Example #2:** We now apply our method to a set of observed data, the 1000-point laser

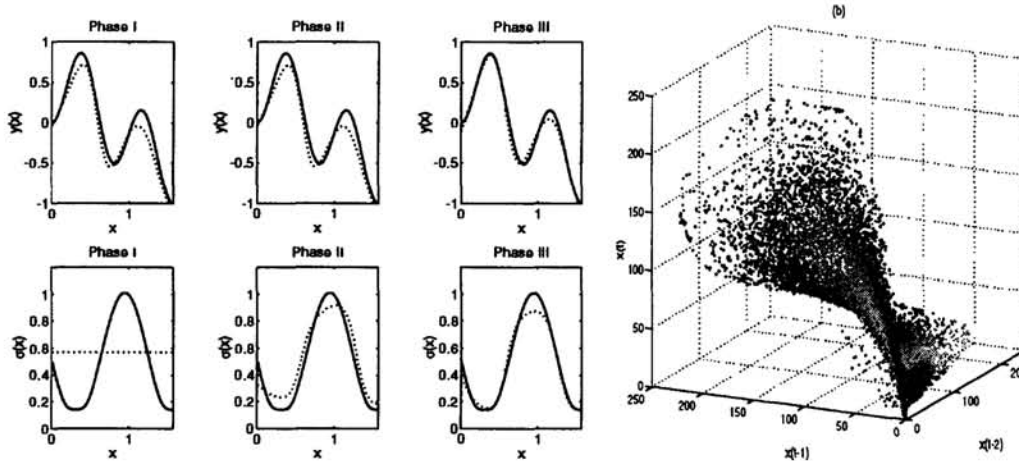

Figure 2: (a) Example #1: Results after each phase of training. The top row gives the true $y(x)$ (solid line) and network estimate $\hat{y}(x)$ (dotted line); the bottom row gives the true $\sigma^2(x)$ (solid line) and network estimate $\hat{\sigma}^2(x)$ (dotted line). (b) Example #2: state-space embedding of laser data (evaluation set) using linear grey-scaling of $0.50$ (lightest) $< \hat{\sigma}(\mathbf{x}_t) < 6.92$ (darkest). See text for details.

intensity series from the Santa Fe competition.[3] Since our method is based on the network's *observed* errors, the predicted error $\hat{\sigma}^2(\mathbf{x})$ actually represents the sum of the underlying system noise, characterized by $\sigma^2(\mathbf{x})$, and the model misspecification. Here, since we know the system noise is roughly uniform 8-bit sampling resolution quantization error, we can apply our method to evaluate the local quality of the manifold approximation.[4]

The prediction task is easier if we have more points that lie on the manifold, thus better constraining its shape. In the competition, Sauer (1994) upsampled the 1000 available data points with an FFT method by a factor of 32. This does not change the effective sampling rate, but it "fills in" more points, more precisely defining the manifold. We use the same upsampling trick (without filtered embedding), and obtain 31200 full $(\mathbf{x}, d)$ patterns for learning. We apply the three-phase approach described above for the simple network of Figure 1 with 25 inputs (corresponding to 25 past values), 12 hidden units feeding the $\hat{y}$-unit, and a liberal 30 hidden units feeding the $\hat{\sigma}^2$-unit (since we are uncertain as to the complexity of $\sigma^2(\mathbf{x})$ for this dataset). We use $\eta = 10^{-7}$ for Phase I and $\eta = 10^{-10}$ for Phases II and III. Since we know the quantization error is $\pm 0.5$, error estimates less than this are meaningless. Therefore, we enforce a minimum value of $\sigma^2(\mathbf{x}) = 0.25$ (the quantization error squared) on the squared errors in Phases II and III.

Table 2:  Results for Example #2 (See Table 1 caption for definitions).

| row | | Training | $(N = 975)$ | Evaluation | $(N = 23,950)$ |
|---|---|---|---|---|---|
| | | $E_{NMS}$ | Mean cost | $E_{NMS}$ | Mean cost |
| 1 | Phase I | 0.00125 | 1.941 | 0.0156 | 7.213 |
| 2 | Phase II | 0.00125 | 1.939 | 0.0156 | 5.628 |
| 3 | Phase III | 0.00132 | 1.725 | 0.0139 | 5.564 |
| | | $\rho$ | | $\rho$ | |
| 4 | $\rho(\hat{\sigma}(x)$, residual errors) | 0.557 | | 0.366 | |
| | | 1 std | 2 std | 1 std | 2 std |
| 5 | % of errors $< \hat{\sigma}(x); 2\hat{\sigma}(x)$ | 69.4 | 94.9 | 63.1 | 88.0 |
| 6 | (*exact Gaussian*) | 68.3 | 95.4 | 68.3 | 95.4 |

Results are given in Table 2 for patterns generated from the available 1000 points and 24,000 additional points used for evaluation. Even though we have used a Gaussian error model, we know the distribution of errors is not Gaussian. This is reflected in rows 5 and 6, where the training data is modeled as having a Gaussian CTD but the evaluation values become considerably distorted from an exact Gaussian. Again, however, not only do we obtain significant predictability of the errors, but the method *also reduces the squared-error measure obtained in Phase I*. We can use the estimated error to characterize the quality of the manifold approximation on 24,000 post-training evaluation points, as illustrated in Figure 2b. The manifold approximation is poorer (darker) for higher predicted values of $x_t$ and for values nearer the edge of the manifold. Note the dark-grey (high-error) vertical streak leaving the origin and dark points to its left which represent patterns involving sudden changes in oscillation intensity.

## 3  Discussion

Since we are in effect approximating two functions simultaneously, we can apply many of the existing variations for improving function approximation designed for networks learning only $\hat{y}(\mathbf{x})$. For example, when using limited amounts of data, especially if it is noisy, the particular split of data into training and cross-validation sets we use introduces significant variation in the resulting $\hat{y}(\mathbf{x})$ due to overfitting, as demonstrated on financial data by Weigend & LeBaron (1994). If we want to estimate local error bars, not only must we fear overfitting $\hat{y}(\mathbf{x})$, but we must also be concerned with overfitting $\hat{\sigma}^2(\mathbf{x})$. If the standard method of stopping at the minimum of an appropriate cross-validation set does not suffice for a given problem, it is straightforward to employ the usual anti-overfitting weaponry (smooth $\hat{\sigma}^2$ as a function of $\mathbf{x}$, pruning, weight-elimination, etc.). Furthermore, we can bootstrap over our available dataset and create multiple composite networks, averaging their predictions for both $\hat{y}(\mathbf{x})$ and $\hat{\sigma}^2(\mathbf{x})$. Additionally, to incorporate prior information in a Bayesian framework as a form of regularization, Wolpert (personal communication, 1994) suggests finding the maximum *a posteriori* (instead of maximum-likelihood) conditional mean and variance using the same interpretation of the network outputs.

In summary, we start with the maximum-likelihood principle and arrive at error estimates that vary with location in the input space. These local error estimates incorporate both underlying system noise and model misspecification. We have provided a computer-generated example to demonstrate the ease with which accurate error bars can be learned. We have also provided an example with real-world data in which the underlying system noise is small, uniform quantization error to demonstrate how the method can be used

to characterize the local quality of the regression model. A significant feature of this method is its weighted-regression effect, which complicates learning by introducing local minima but can be potentially beneficial in constructing a more robust model with improved generalization abilities. In the framework presented, for any problem we must assume a specific parameterized CTD then add one auxiliary output unit for each higher moment of the CTD we wish to estimate locally. Here we have demonstrated the Gaussian case with a location parameter (conditional mean) and a scale parameter (local error bar) for a scalar output variable. The extension to multiple output variables is clear and allows a full covariance matrix to be used for weighted regression, including the cross-correlation between multiple targets.

## Acknowledgments

This work is supported by the National Science Foundation under Grant No. RIA ECS-9309786 and by a Graduate Fellowship from the Office of Naval Research. We would like to thank Chris Bishop, Wray Buntine, Don Hush, Steve Nowlan, Barak Pearlmutter, Dave Rumelhart, and Dave Wolpert for helpful discussions.

## Footnotes

*http://www.cs.colorado.edu/~andreas/Home.html.

[1] The case of a single Gaussian to represent a unimodal distribution can also been generalized to a mixture of several Gaussians that allows the modeling of multimodal distributions (Bishop, 1994).

[2]Further details: all inputs are scaled to zero mean and unit variance. All initial weights feeding into hidden units are drawn from a uniform distribution between $-1/i$ and $1/i$ where $i$ is the number of incoming connections. All initial weights feeding into $\hat{y}$ or $\hat{\sigma}^2$ are drawn from a uniform distribution between $-s/i$ and $s/i$ where $s$ is the standard deviation of the (overall) target distribution. No momentum is used, and all weight updates are averaged over the forward passes of 20 patterns.

[3]The data set and several predictions and characterizations are described in the volume edited by Weigend & Gershenfeld (1994). The data is available by anonymous ftp at `ftp.cs.colorado.edu` in `/pub/Time-Series/SantaFe` as `A.dat`. See also `http://www.cs.colorado.edu/Time-Series/TSWelcome.html` for further analyses of this and other time series data sets.

[4]When we make a single-step prediction where the manifold approximation is poor, we have little confidence making iterated predictions based on that predicted value. However, if we know we are in a low-error region, we can have increased confidence in iterated predictions that involve our current prediction.

## References

C. Bishop. (1994) "Mixture Density Networks." Neural Computing Research Group Report NCRG/4288, Department of Computer Science, Aston University, Birmingham, UK.

W.L. Buntine and A.S. Weigend. (1991) "Bayesian Backpropagation." *Complex Systems*, **5**: 603–643.

A.M. Fraser and A. Dimitriadis. (1994) "Forecasting Probability Densities Using Hidden Markov Models with Mixed States." In *Time Series Prediction: Forecasting the Future and Understanding the Past*, A.S. Weigend and N.A. Gershenfeld, eds., Addison-Wesley, pp. 265–282.

P.T. Kazlas and A.S. Weigend. (1995) "Direct Multi-Step Time Series Prediction Using TD($\lambda$)." In *Advances in Neural Information Processing Systems 7 (NIPS*94, this volume)*. San Francisco, CA: Morgan Kaufmann.

D.E. Rumelhart, R. Durbin, R. Golden, and Y. Chauvin. (1995) "Backpropagation: The Basic Theory." In *Backpropagation: Theory, Architectures and Applications*, Y. Chauvin and D.E. Rumelhart, eds., Lawrence Erlbaum, pp. 1–34.

T. Sauer. (1994) "Time Series Prediction by Using Delay Coordinate Embedding." In *Time Series Prediction: Forecasting the Future and Understanding the Past*, A.S. Weigend and N.A. Gershenfeld, eds., Addison-Wesley, pp. 175-193.

A.N. Srivastava and A.S. Weigend. (1994) "Computing the Probability Density in Connectionist Regression." In *Proceedings of the IEEE International Conference on Neural Networks (IEEE–ICNN'94), Orlando, FL*, p. 3786–3789. IEEE-Press.

A.S. Weigend and N.A. Gershenfeld, eds. (1994) *Time Series Prediction: Forecasting the Future and Understanding the Past*. Addison-Wesley.

A.S. Weigend and B. LeBaron. (1994) "Evaluating Neural Network Predictors by Bootstrapping." In *Proceedings of the International Conference on Neural Information Processing (ICONIP'94), Seoul, Korea*, pp. 1207–1212.

A.S. Weigend and D.A. Nix. (1994) "Predictions with Confidence Intervals (Local Error Bars)." In *Proceedings of the International Conference on Neural Information Processing (ICONIP'94), Seoul, Korea*, p. 847–852.